# Approximability of Probability Distributions

**Alina Beygelzimer**[*]
IBM T. J. Watson Research Center
Hawthorne, NY 10532
beygel@cs.rochester.edu

**Irina Rish**
IBM T. J. Watson Research Center
Hawthorne, NY 10532
rish@us.ibm.com

## Abstract

We consider the question of how well a given distribution can be approximated with probabilistic graphical models. We introduce a new parameter, *effective treewidth*, that captures the degree of approximability as a tradeoff between the accuracy and the complexity of approximation. We present a simple approach to analyzing achievable tradeoffs that exploits the threshold behavior of monotone graph properties, and provide experimental results that support the approach.

## 1  Introduction

One of the major concerns in probabilistic reasoning using graphical models, such as Bayesian networks, is the computational complexity of inference. In general, probabilistic inference is NP-hard and a typical approach to handling this complexity is to use an approximate inference algorithm that trades accuracy for efficiency. This leads to the following question: How can we distinguish between distributions that are easy to approximate and those that are hard? More generally, how can we characterize the inherent degree of distribution's complexity, i.e. its *approximability*?

These questions also arise in the context of learning probabilistic graphical models from data. Note that traditional model selection criteria, such as BIC/MDL, aim at fitting the data well and minimizing the *representation* complexity of the learned model (i.e., the total number of parameters). However, as demonstrated in [2], such criteria are unable to capture the *inference* complexity: two models that have similar representation complexity and fit data equally well may have quite different graph structures, making one model *exponentially* slower for inference than the other. Thus, our goal is to develop learning algorithms that can find good trade-offs between accuracy of a model and its inference complexity.

Commonly used exact inference algorithms, such as the junction tree algorithm [12], or closely related variable-elimination techniques [6], essentially triangulate the graph, and their complexity is exponential in the size of largest clique induced during triangulation (parameter known as *treewidth*). Generally, it can be shown that (in some precise sense) *any* scheme for belief updating based on local calculations must contain a hidden triangulation [10]. Thus the treewidth arises as a natural measure of inference complexity in graphical models.

---

[*]The work was done while the author was at the Department of Computer Science, University of Rochester.

Intuitively, a probability distribution is approximable, or *easy*, if it is *close* to a distribution represented by an efficient, low-treewidth graphical model. We use the Kullback-Leibler divergence $d_{KL}$ as a measure of closeness.[1] The following example explains our intuition behind approximable vs. nonapproximable distributions.

**Motivating Example**  Consider the parity function on $n$ binary random variables $\{X_1, \ldots, X_n\}$, and let our target distribution $P$ be the uniform distribution on the values to which it assigns 1 (i.e., on $n$-bit strings with an odd number of 1s). It is easy to see that any approximation $Q$ that decomposes over a network whose moralized graph misses at least one edge, is precisely as inaccurate as the one that assumes all variables to be independent (i.e., has no edges).

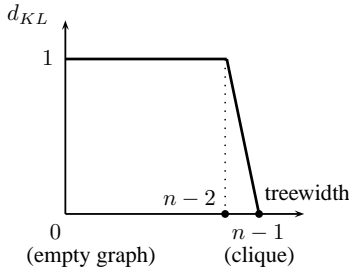

This follows from the fact that the probability distribution induced on any proper subset of the variables is uniform, and thus for any subset $\{X_{i_1}, \ldots, X_{i_k}\}$ of $k < n$ variables, $P(X_{i_1} \mid X_{i_2}, \ldots, X_{i_k}) = P(X_{i_1})$, uniform on $\{0,1\}$. It is then readily seen that $\sum_{\mathbf{x}} P(\mathbf{x}) \log Q(\mathbf{x}) = 2^{-(n-1)} \sum_{\mathbf{x}:P(\mathbf{x})>0} \log \prod_{i=1}^{n} Q(x_i \mid x_{i_1}, \ldots, x_{i_r}) = \log \prod_{i=1}^{n} Q(x_i) = \log 2^{-n} = -n$,[2] and $d_{KL}(P,Q) = -H(P)+n = 1$ since $H(P) = n-1$. Thus, unless we can afford the complexity of the complete graph, there is *absolutely* no sense (i.e., absolutely no gain in accuracy and a potentially exponential loss of efficiency) in using a model more complex than the empty graph (i.e., $n$ isolated nodes with no edges). This intuitively captures what we mean by a nonapproximable distribution.

On the other hand, one can easily construct a distribution with large weak dependencies such that representing this distribution exactly requires a network with large treewidth; however, if we are willing to sacrifice just a bit of accuracy, we get a very simple model. For example, consider a distribution $P(\{X_1, \ldots, X_n\})$ in which variables $X_1, \ldots, X_{n-1}$ are independent and uniformly distributed; if all $X_1, \ldots, X_{n-1}$ are true, $X_n$ is true with probability 1 (and false with probability 0); otherwise $X_n$ is true with probability 1/2 (regardless of the values of $X_1, \ldots, X_{n-1}$). The network yielding zero KL-divergence is the $n$-node clique (after moralization). Tolerating KL-divergence $2^{-(n-1)}$ (i.e., exponentially vanishing with $n$) allows us to use an exponentially more efficient model for $P$ (namely, the empty graph).

The following questions naturally arise: If we tolerate a certain inaccuracy, what is the best inference complexity we can hope to achieve? Or, what is the best achievable approximation accuracy given a constraint on the complexity (i.e., a bound on the treewidth)? The tradeoff between the complexity and accuracy is monotonic; however, it may be far from linear. The goal is to exploit these nonlinearities in choosing the best available tradeoff.

Our analysis of accuracy vs. complexity trade-offs is based on the results from random graph theory which suggest that graph properties monotone in edge addition (e.g., such as graph connectivity) appear rather suddenly: the transition from the property being very unlikely to it being very likely occurs during a small change of the edge probability $p$ (density) in the random graph [7, 8].

This paper makes the following contributions. First, we show that both important properties of random graphical models, the property of "being efficient" (i.e., having treewidth at most some fixed integer $k$) and the property of "being accurate" (i.e., being at distance at most some $\delta$ from the target distribution), are monotone and demonstrate a threshold behavior, giving us two families of threshold curves parameterized by $k$ and by $\delta$, respectively. Second, we introduce the notion of *effective treewidth* $k(\delta)$, which denotes the smallest

achievable treewidth $k$ given a constraint $\delta$ on KL-divergence (error) from the target (we also introduce a notion of $\epsilon$-achievable $k(\delta)$ which requires at least $\epsilon$-fraction of models in a given set to achieve treewidth $k$ and error $\delta$). The effective treewidth captures the approximability of the distribution, and is determined by relative position of the threshold curves, an inherent property of the target distribution. Finally, we provide an efficient sampling-based approach that actually finds a model achieving $k(\delta)$ with high probability. We estimate the threshold curves and, using their relative position, identify a class of treewidth-bounded models such that the models in the class are *still* simple, yet this class *already* contains (with high probability) a sufficiently good approximations to the target distribution (otherwise, we suggest that the distribution is inherently hard to approximate).

## 2 Preliminaries and Related Work

Let $P$ be a probability distribution on $n$ discrete random variables $X_1, X_2, \ldots, X_n$. A *Bayesian network* exploits the independences among the $X_i$ to provide a compact representation of $P$ as a product of low-order conditional probability distributions. The independences are encoded by a directed acyclic graph (DAG) $G$ with nodes corresponding to $X_1, X_2, \ldots, X_n$ and edges representing direct dependencies. Each $X_i$ is independent of its non-descendants given its parents in the graph [12]. The dependencies are quantified by associating each node $X_i$ with a local conditional probability distribution $P_B(X_i \mid \Pi_i)$, where $\Pi_i$ is the set of parents of $X_i$ in $G$. The joint probability distribution encoded by $B$ is given by the product $P_B(X_1, \ldots, X_n) = \prod_{i=1}^{n} P_B(X_i \mid \Pi_i)$. We say that a distribution $P$ *decomposes* over a DAG $G$ if there exist local conditional probability distributions corresponding to $G$ such that $P$ can be written in such a form.

In general, exact probabilistic inference in Bayesian networks is NP-hard. For singly-connected networks (i.e., networks with no undirected cycles), there is a linear time local belief-propagation algorithm [12]. In order to use this algorithm in the presence of cycles, one typically constructs a *junction tree* of the network and runs the algorithm on this tree [12]. Constructing a junction tree involves triangulating the graph, i.e., adding edges so that every cycle of length greater than three has a chord (i.e., an edge between a pair of non-adjacent nodes). Each triangulation corresponds to some order of eliminating variables when summing terms out during inference [6]. Exact inference can then be done in time and space linear in the representation of clique marginals in the junction tree, which is exponential in the size of the largest clique induced during triangulation. This number (minus one) is known as the *width* of a given triangulation. The minimum width over all possible triangulations is called the *treewidth* of the graph. The triangulation procedure is defined for undirected graphs, so we must first make the network undirected while preserving the set of independence assumptions; this can be done by *moralizing* the network, i.e., connecting ("marrying") the parents of every node by a clique and then dropping the direction of all edges.

Given a set of independent samples from $P$, the general goal is to learn a model (a Bayesian network) of this distribution that involves dependencies only on limited subsets of the variables. Restricting the size of dependencies controls both overfitting and the complexity of inference in the resulting model. The samples are in the form of tuples $\langle x_1, \ldots, x_n \rangle$ each corresponding to a particular assignment $\langle X_1 = x_1, \ldots, X_n = x_n \rangle$. Given a target distribution $P(\mathbf{X})$ and an approximation $Q(\mathbf{X})$, the *information divergence* (or Kullback-Leibler distance) between $P$ and $Q$ is defined as $d_{KL}(P, Q) = \sum_{\mathbf{x}} P(\mathbf{x}) \log \frac{P(\mathbf{x})}{Q(\mathbf{x})}$, where $\mathbf{x}$ ranges over all possible assignments to the variables in $\mathbf{X}$ (See [5].) Notice that $d_{KL}(P, Q)$ is not necessarily symmetric.

A natural way of controlling the complexity of the learned model is to limit ourselves to a class of treewidth-bounded networks. Let $\mathcal{D}_k$ denote the class of distributions decomposable on graphs with treewidth at most $k$ ($0 \leq k < n$), with $\mathcal{D}_1$ corresponding to the set of

tree-decomposable distributions. The distribution within $\mathcal{D}_k$ minimizing the information divergence from the target distribution $P$ is called the *projection* of $P$ onto $\mathcal{D}_k$. Again, if $P$ is the empirical distribution, then this is also the distribution within $\mathcal{D}_k$ maximizing the likelihood of observing the data.

**Learning bounded-treewidth models**    Chow and Liu [4] showed how to find a projection onto the set of tree-decomposable distributions. For a fixed tree $T$, the projection of $P$ onto the set of $T$-decomposable distributions is uniquely given by the distribution in which the conditional probabilities along the edges of $T$ coincide with those computed from $P$. Hence the tree yielding the closest projection is simply given by any maximum weight spanning tree, where the edge weight is the mutual information between the corresponding variables. Notice that candidate spanning trees can be compared without any knowledge of $P$ beyond that given by pairwise statistics. The tree can be efficiently found using any of the well known algorithms. The additive decomposition of $d_{KL}$ used in the proof, can be easily extended to "wider" networks. Fix a network structure $G$, and let $Q$ be a distribution decomposable over $G$. Then

$$d_{KL}(P, Q) = \sum_{\mathbf{x}} P(\mathbf{x}) \log \frac{P(\mathbf{x})}{Q(\mathbf{x})} = -\sum_{i=1}^{n} \sum_{x_i, \pi_i} P(x_i, \pi_i) \log Q(x_i \mid \pi_i) - H(P),$$

where $\pi_i$ ranges over all possible values of $\Pi_i$. If $P$ is the empirical distribution induced by the given sample of size $N$ (i.e., defined by frequencies of events in the sample), then the first term can be shown to be $-LL(Q)/N$.[3] Thus minimizing $d_{KL}(P, Q)$ is equivalent to maximizing the log likelihood $LL(Q)$.

Standard arguments (see, for example, [12]) show that the first term is maximized by forcing all conditional probabilities $Q(x_i \mid \pi_i)$ to coincide with those computed from $P$. If $P$ is the empirical distribution, this means forcing the parameters to be the corresponding relative frequencies in the sample. Hence if $G$ is fixed, the projection onto the set of $G$-decomposable distributions is uniquely defined, and we will identify $G$ with this projection (ignoring some notational abuse). It remains, of course, to find $G$ that is the closest to $P$ among all DAGs in some treewidth-bounded class $\mathcal{D}_k$. As observed by Höffgen [9], the problem readily reduces to the minimum-weight hypertree problem. The reverse reduction is not known, so the NP-hardness of the hypertree problem does not imply the hardness of the learning problem. Srebro [13] showed that a similar *undirected* decomposition holds for bounded treewidth Markov networks (probabilistic models that use undirected graphs to represent dependencies). He showed that the learning problem is *equivalent* to finding a minimum-weight undirected hypertree, and so is NP-hard. It is important to note that Srebro [13] considered approximation in the context of density estimation rather than model selection, thus the choice of $k$ is directly driven by the size of the sample space; the only rationale for limiting the class of hypothesis distributions is to prevent overfitting. With an infinite amount of data, they would learn a clique, since adding edges would always decrease the divergence. Our goal, on the other hand, is to find the most appropriate treewidth-bounded class onto which to project the distribution.

**Threshold behavior of random graphs**    We use the model of random directed acyclic graphs (DAGs) defined by Barak and Erdős [1]. Consider the probability space $G(n, p)$ of random undirected graphs on $n$ nodes with edge probability $p$ (i.e., every pair of nodes is connected with probability $p$, independently of every other pair). Let $G_{n,p}$ stand for a random graph from this probability space. We will also occasionally use $G_{n,m}$ to denote a graph chosen randomly from among all graphs with $n$ nodes and $m$ edges. When $p = m/\binom{n}{2}$, the two models are practically identical. A random DAG in the Barak-Erdős model is obtained from $G_{n,p}$ by orienting the edges according to the ordering of vertices, i.e., all edges are directed from higher to lower indexed vertices.

A *graph property* $\mathcal{P}$ is naturally associated with the set of graphs having $\mathcal{P}$. A property is *monotone increasing* if it is preserved under edge addition: If a graph $G$ satisfies the property, then every graph on the same set of nodes containing $G$ as a subgraph must satisfy it as well. It is easy to see (and intuitively clear) that if $\mathcal{P}$ is a monotone increasing property then the probability that $G_{n,p}$ satisfies $\mathcal{P}$ is a non-decreasing function of $p$. A *monotone decreasing* property is defined similarly. For example, the property of having treewidth at most some fixed integer $k$ is monotone decreasing: adding edges can only increase the treewidth. The theory of random graphs was initiated by Erdős and Rényi [7], and one of the main observations they made was that many natural monotone properties appear rather suddenly, i.e., as we increase $p$, there is a sharp transition from a property being very unlikely to it being very likely. Friedgut [8] proved that *every* monotone graph property of undirected graphs has such a threshold behavior. Random DAGs (corresponding to random partially ordered sets) have received less attention then random undirected graphs, partially because of the additional structure that prevents the completely independent choice of edges. Nonetheless, many properties of random DAGs were also shown to have threshold functions. (See, for example, [3] and references therein.) However, we are not aware of any general result for random DAGs analogous to that of Friedgut [8].

## 3  Formalization

First we introduce two properties of networks essential for the rest of the paper.

**Accuracy**   Recall that the information divergence of a given DAG $G$ from the target distribution $P$ is given by $d_{KL}(P,G) = W(G) - H(P)$, where $W(G) = -\sum_{i=1}^{n}\sum_{x_i,\pi_i} P(x_i,\pi_i) \log P(x_i \mid \pi_i)$. (In our case, $P$ is the empirical distribution induced by the given sample $S$ of size $N$. As mentioned before, $W(G) = -LL(G)/N \geq 0$.) Fix a distance parameter $\delta > 0$, and consider the property $\mathcal{P}_\delta$ of $n$-node DAGs of having $W(G) \leq \delta$. Notice that $\mathcal{P}_\delta$ is monotone increasing: Adding edges to a graph can only bring the graph closer to the target distribution, since any distribution decomposable on the original graph is also decomposable on the augmented one. Thus if $G$ is a subgraph of $G'$, then $W(G) \leq \delta$ only if $W(G') \leq \delta$.

**Complexity**   Fix an integer $k$, and consider the property of $n$-node DAGs of having treewidth of their moralized graph at most $k$. Call this property $\mathcal{P}_k$ and note that it is a structural property of a DAG, which does *not* depend on the target distribution and its projection onto the DAG. It is also a monotone decreasing property, since if a graph has treewidth at most $k$, then certainly any of its subgraphs does.

Recall that we identify each graph with the projection of the target distribution onto the graph. We call a pair $(k,\delta)$ *achievable* for a distribution $P$, if there exists a distribution $Q$ decomposable on a graph with treewidth at most $k$ such that $d_{KL}(P,Q) \leq \delta$. The *effective treewidth* of $P$, with respect to a given $\delta$, is defined as the smallest $k(\delta)$ such that the pair $(k,\delta)$ is achievable, i.e., if all distributions at distance at most $\delta$ from $P$ are not decomposable on graphs with treewidth less than $k(\delta)$. This formulation gives the level of inevitable complexity (i.e., treewidth) $k$, given the desired level of accuracy $\delta$. We will also be interested in average-case analogs of these definitions. Fix $\epsilon > 0$. We will say that a pair $(k,\delta)$ is $\epsilon$-*achievable* for $P$ if at least an $\epsilon$-fraction of all DAGs in $\mathcal{D}_k$ certify that $(k,\delta)$ is achievable. Thus we not only care about the existence of an approximation with given $\delta$ and $k$, but also in the *number* of such approximations.

## 4  Main Idea

Consider, for each treewidth bound $k$, the curve given by $\mu_k(p) = \mathbf{Pr}[\text{width}(G_{n,p}) \leq k]$, and let $p_k$ be such that $\mu_k(p_k) = 1/2 + \epsilon$, where $0 < \epsilon < \frac{1}{2}$ is some fixed constant. Similarly, for $\delta > 0$, define the curve $\mu_\delta(p) = \mathbf{Pr}[W(G_{n,p}) \leq \delta]$, and let $p_\delta$ be the critical value of $p$ given by $\mu_\delta(p_\delta) = 1/2$.

For reasons that will become clear in a moment, our goal will be to find, for each feasible treewidth $k$, the value of $\delta$ such that $p_\delta = p_k$. To find each $p_k$, the algorithm will simply do a binary search on the interval $(0, 1)$: For the current value of edge probability $p$, the algorithm estimates $\mu_k(G_{n,p})$ by random sampling and branches according to the estimate. The search is continued until $p$ gets sufficiently close to satisfying $\mu_k(G_{n,p}) = 1/2 + \epsilon$. To estimate $\mu_k(G_{n,p})$ within an additive error $\rho$ with probability at least $1 - \gamma$, the algorithm samples $m = \frac{\ln(2/\gamma)}{2\rho^2}$ independent copies of $G_{n,p}$, and outputs the average value of the 0/1 random variable indicating whether the treewidth of the sampled DAG is at most $k$. The analysis is just a straightforward application of the Chernoff Bound. Note that the values related to treewidth are independent of the target distribution and can be precomputed offline. To find $\delta = \delta(k)$ for a given value of $k$, the algorithm computes the values of $W(G_{n,p_k})$ for the $m$ sampled random DAGs in $G(n, p_k)$, orders them and chooses the median. Each pair $(k, \delta)$ gives a point on the threshold curve. We know that at least a $(1/2 + \epsilon)$-fraction of the DAGs in $G(n, p_k)$ satisfy $\mathcal{P}_k$. On the other hand, at least half of them satisfy $\mathcal{P}_\delta$, and thus at least an $\epsilon$-fraction satisfies both. Moreover, there is a very simple probabilistic algorithm for finding a model realizing the tradeoff: We just need to sample $O(1/\epsilon)$ DAGs in $G(n, p_k)$ and choose the closest one. Clearly we are overcounting, since the same DAGs may contribute to both probabilities; however not absurdly, since intuitively the graphs in $G(n, p_k)$ with small treewidth will not fit the distribution better than the ones with larger treewidth.

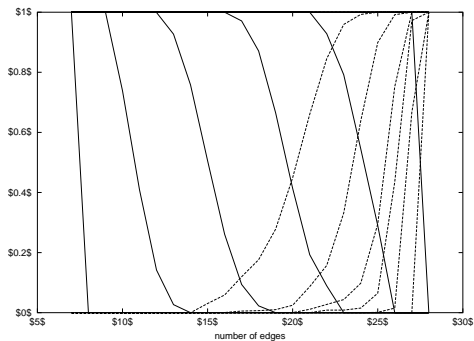

Figure 1: Threshold curves for a 3-wise independent distribution on 8 random variables (using a construction from [11]).

A small example should help make the goals clear. A distribution is called *k-wise independent* if any subset of $k$ variables is mutually independent (however, there may exist dependencies on larger subsets). Figure 1 shows the curves for a 3-wise independent distribution on 8 random variables. We can hardly expect graphs with treewidth at most 2 to do well on this distribution, since all triples are independent, and their marginals do not reveal any higher-order structure; as we will see this is indeed the case. The $x$-axis in Figure 1 corresponds to the number of edges $m$, the $y$-axis denotes the probability that $G_{n,m}$ satisfies the property corresponding to a given curve. The monotone decreasing curves correspond to the properties $\mathcal{P}_k$ for $k = \{1, \ldots, 6\}$ (from left to right respectively). For $k = 7$, the curve is just $\mu_m(\mathcal{P}_k) = 1$. The monotone increasing curves correspond to the property of having $d_{KL}$ at most $\delta$. The leftmost curve is for $\delta = 0.07$, and it decreases by 0.01 as we go from left to right; the smaller $\delta$, the higher the quality of approximation, thus the smaller the probability of attaining it. The empty graph (treewidth 0) had divergence 0.073. As $m$ increases, the probability of having small treewidth decreases, while the probability of getting close to the target increases. (Since $n$ is small, we computed the divergence exactly.) As the random graph evolves, we want to capture the moment when the first probability is *still* high, while the second is *already* high. As expected, graphs with treewidth at most 2 are as inaccurate as the empty graph since all triples are independent. Given the desired level of closeness $\delta$, we want to find the smallest treewidth $k$ such that the corresponding curves meet above some cut-off probability. For example, to get within $d_{KL}$ at most 0.7, we may suggest, say, projecting onto graphs with treewidth 4 (cutting at 0.4). The cut-off value determines the efficiency of finding a model with such $k$ and $\delta$ (see discussion above).

**Estimating** $d_{KL}$  Fix a bounded-treewidth DAG $G$. Let the target distribution be the empirical distribution $P$ induced by a given sample. Recall that $d_{KL}(P,G)$ decomposes into sum of conditional entropies induced by $G$ (minus the entropy of $P$). Höffgen [9] showed how to estimate these conditional entropies with any fixed additive precision $\rho$ using polynomially many samples. More precisely, he showed that a sample of size $m = m(\gamma, \rho) = O((\frac{n}{\rho})^2 \log^2 \frac{n}{\rho} \log \frac{n^{k+1}}{\gamma})$ suffices to obtain good estimations of all induced conditional entropies with probability at least $1 - \gamma$, which in turn suffices to estimate $d_{KL}(P,G)$ with the additive precision $\rho$.

**Estimating Treewidth**  We, of course, will not attempt to compute the treewidth of the randomly generated graphs exactly. The problem is NP-hard [4]. In practice, people often use heuristics (based, for example, on eliminating vertices in the order of maximum cardinality, minimum degree, or minimum separating vertex set). There are no theoretical guarantees in general, but heuristics tend to perform reasonably well: used in combination with various lower bound techniques, they can often pin down the treewidth to a small range, or even identify it exactly [5]. We stress that the values related to treewidth are independent of the target distribution and can be precomputed.

## 5   Experimental Results

We tested the approach presented in the paper on distributions ancestrally sampled from real-life medical networks commonly used for benchmarking. The experiments support the following conclusions: the properties capturing the complexity and accuracy of a model indeed demonstrate a threshold behavior, which can be exploited in determining the best tradeoff for the given distribution; the simple approach based on generating random graphs and using them to approximate the thresholds is indeed capable of capturing the effective width of a distribution. Due to page limit, we discuss an application of the method to a single network known as ALARM (originating from anesthesia monitoring).

The network has $37$ nodes, $46$ directed edges, $19$ additional undirected edges induced by moralization; the treewidth is $4$. A sample of size $N = 10^4$ was generated using ancestral sampling, inducing the empirical distribution with support on $5570$ unique variable assignments. The entropy of the empirical distribution $P$ was $9.6$ (maximum possible entropy for a $5570$-point distribution is $12.4$). Figure 2 shows the curve illustrating the (estimated) tradeoffs available for $P$. For each treewidth bound $k$, the curves gives an estimate of the best achievable value of $W = d_{KL} - H(P)$. (Recall that $LL = -N \cdot W$.)

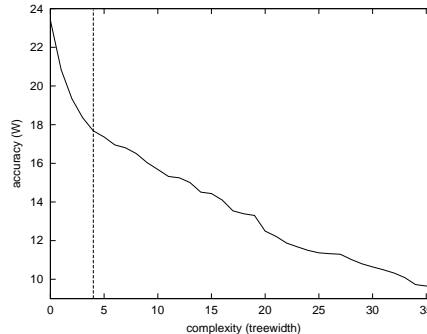

Figure 2: Tradeoff curve for ALARM

The estimate is based on generating $400$ random DAGs with $37$ nodes and $m$ edges, for every possible $m$. Several points on the curve are worthy of note. The upper-left point $(0, 23.4)$ corresponds to the model that assumes all $37$ variables to be independent. On the other extreme, the lower-right point $(36, 0)$ corresponds to the clique on $37$ nodes, which of course can model $P$ perfectly, but with exponential complexity. The closer the area under the curve to zero, the easier the distribution (in the sense discussed in this paper). Here we see that the highest gain in accuracy from allowing the model to be more complex occurs up to treewidth 4, less so 5 and 6; by further increasing the treewidth we do not gain much in accuracy. We succeed in reconstructing the width in the sense that the distribution was

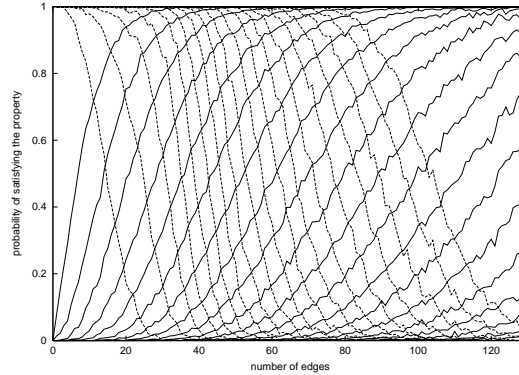

Figure 3: Threshold curves for ALARM

simulated from a treewidth-4 model.[6] Such tradeoff curves are similar to commonly used ROC (Receiver Operating Characteristic) curves; the techniques for finding the cutoff value in ROC curves can be used here as well. Instead of plotting the best achievable distance, we can plot the best distance achievable by at least an $\epsilon$-fraction of models in the class, parameterizing the tradeoff curve by $\epsilon$. Figure 3 shows the threshold curves. The axes have the same meaning as in Figure 1. Varying sample size and the number of randomly generated DAGs does not change the behavior of the curves in any meaningful way; not surprisingly, increasing these parameters results in smoother curves.

## Footnotes

[1]Note that minimizing $d_{KL}$ from the empirical distribution (induced by a given set of samples) also corresponds to maximizing the likelihood of observed data.

[2]The second to last equality is due to the well-known fact that $d_{KL}(P,Q)$ is minimized by forcing the conditional probabilities of $Q$ to coincide with those computed from $P$.

[3]Since the true distribution $P$ is given only by the sample, we let $P$ also denote the empirical distribution induced by the sample, ignoring some abuse of notation.

[4]If $k$ is fixed, the problem of determining whether a graph has treewidth $k$ has a linear time algorithm. As typical, the bound contains a large hidden constant with $k$ in the exponent, making the algorithm hardly applicable in practice. There is a number of constant-factor approximations with an exponential dependence on $k$, and a polynomial-time $O(\log k)$-factor approximation. No polynomial-time constant-factor approximation is known.

[5]Although one can construct graphs for which they produce solutions that are arbitrarily far from optimal.

[6]Note, however, that it does not imply that the empirical distribution itself decomposes on a treewidth-4 model. The simplest example of this is when the true distribution is uniform.

## References

[1] A. Barak and P. Erdős. On the maximal number of strongly independent vertices in a random acyclic directed graph. *SIAM J. Algebraic and Discrete Methods*, 5:508–514, 1984.

[2] A. Beygelzimer and I. Rish. Inference complexity as a model-selection criterion for learning bayesian networks. In *Proceedings of the Eighth International Conference on Principles of Knowledge Representation and Reasoning (KR2002), Toulouse, France*, 2002.

[3] B. Bollobás and G. Brightwell. The structure of random graph orders. *SIAM J. Discrete Mathematics*, 10(2):318–335, 1997.

[4] C. Chow and C. Liu. Approximating discrete probability distributions with dependence trees. *IEEE Trans. on Inf. Theory*, 14:462–467, 1968.

[5] T. Cover and J. Thomas. *Elements of information theory*. John Wiley & Sons Inc., New York, 1991. A Wiley-Interscience Publication.

[6] R. Dechter. *Bucket elimination: A unifying framework for probabilistic reasoning*. In M. I. Jordan (Ed.), Learning in Graphical Models, Kluwer Academic Press, 1998.

[7] P. Erdős and A. Rényi. On the evolution of random graphs. *Bull. Inst. Internat. Statist.*, 38:343–347, 1961.

[8] E. Friedgut and G. Kalai. Every monotone graph property has a sharp threshold. *Proceedings of the American Mathematical Society*, 124(10):2993–3002, 1996.

[9] K. Höffgen. Learning and robust learning of product distributions. In *Proceedings of the 6th Annual Workshop on Computational Learning Theory*, pages 77–83, 1993.

[10] F. V. Jensen and F. Jensen. Optimal junction trees. In *Proc. Tenth Conference on Uncertainty and AI (UAI)*, 1994.

[11] J. Naor and M. Naor. Small-bias probability spaces: Efficient constructions and applications. In *Proc. of the 22nd ACM Symposium on Theory of Computing (STOC)*, pages 213–223, 1990.

[12] J. Pearl. *Probabilistic Reasoning in Intelligent Systems: Networks of Plausible Inference*. Morgan Kaufmann Publishers, 1988.

[13] N. Srebro. Maximum likelihood bounded Tree-Width markov networks. In *Proceedings of the 17th Conference on Uncertainty in AI (UAI)*, pages 504–511, 2001.

